# Neural Networks for Density Estimation

**Malik Magdon-Ismail***
magdon@cco.caltech.edu
Caltech Learning Systems Group
Department of Electrical Engineering
California Institute of Technology
136-93 Pasadena, CA, 91125

**Amir Atiya**
amir@deep.caltech.edu
Caltech Learning Systems Group
Department of Electrical Engineering
California Institute of Technology
136-93 Pasadena, CA, 91125

## Abstract

We introduce two new techniques for density estimation. Our approach poses the problem as a supervised learning task which can be performed using Neural Networks. We introduce a stochastic method for learning the cumulative distribution and an analogous deterministic technique. We demonstrate convergence of our methods both theoretically and experimentally, and provide comparisons with the Parzen estimate. Our theoretical results demonstrate better convergence properties than the Parzen estimate.

## 1 Introduction and Background

A majority of problems in science and engineering have to be modeled in a probabilistic manner. Even if the underlying phenomena are inherently deterministic, the complexity of these phenomena often makes a probabilistic formulation the only feasible approach from the computational point of view. Although quantities such as the mean, the variance, and possibly higher order moments of a random variable have often been sufficient to characterize a particular problem, the quest for higher modeling accuracy, and for more realistic assumptions drives us towards modeling the available random variables using their probability density. This of course leads us to the problem of density estimation (see [6]).

The most common approach for density estimation is the nonparametric approach, where the density is determined according to a formula involving the data points available. The most common non parametric methods are the kernel density estimator, also known as the Parzen window estimator [4] and the $k$-nearest neighbor technique [1]. Non parametric density estimation belongs to the class of ill-posed problems in the sense that small changes in the data can lead to large changes in

the estimated density. Therefore it is important to have methods that are robust to slight changes in the data. For this reason some amount of regularization is needed [7]. This regularization is embedded in the choice of the smoothing parameter (kernel width or $k$). The problem with these non-parametric techniques is their extreme sensitivity to the choice of the smoothing parameter. A wrong choice can lead to either undersmoothing or oversmoothing.

In spite of the importance of the density estimation problem, proposed methods using neural networks have been very sporadic. We propose two new methods for density estimation which can be implemented using multilayer networks. In addition to being able to approximate any function to any given precision, multilayer networks give us the flexibility to choose an error function to suit our application. The methods developed here are based on approximating the distribution function, in contrast to most previous works which focus on approximating the density itself. Straightforward differentiation gives us the estimate of the density function. The distribution function is often useful in its own right - one can directly evaluate quantiles or the probability that the random variable occurs in a particular interval.

One of the techniques is a stochastic algorithm (SLC), and the second is a deterministic technique based on learning the cumulative (SIC). The stochastic technique will generally be smoother on smaller numbers of data points, however, the deterministic technique is faster and applies to more that one dimension. We will present a result on the consistency and the convergence rate of the estimation error for our methods in the univariate case. When the unknown density is bounded and has bounded derivatives up to order $K$, we find that the estimation error is $O((\log \log(N)/N)^{-(1-\frac{1}{K})})$, where $N$ is the number of data points. As a comparison, for the kernel density estimator (with non-negative kernels), the estimation error is $O(N^{-4/5})$, under the assumptions that the unknown density has a square integrable second derivative (see [6]), and that the *optimal* kernel width is used, which is not possible in practice because computing the optimal kernel width requires knowledge of the true density. One can see that for smooth density functions with bounded derivatives, our methods achieve an error rate that approaches $O(N^{-1})$.

## 2   New Density Estimation Techniques

To illustrate our methods, we will use neural networks, but stress that any sufficiently general learning model will do just as well. The network's output will represent an estimate of the distribution function, and its derivative will be an estimate of the density. We will now proceed to a description of the two methods.

### 2.1   SLC (Stochastic Learning of the Cumulative)

Let $x_n \in \mathbf{R}$, $n = 1, ..., N$ be the data points. Let the underlying density be $g(x)$ and its distribution function $G(x) = \int_{-\infty}^{x} g(t)dt$. Let the neural network output be $H(x, w)$, where $w$ represents the set of weights of the network. Ideally, after training the neural network, we would like to have $H(x, w) = G(x)$. It can easily be shown that the density of the random variable $G(x)$ ($x$ being generated according to $g(x)$) is uniform in $[0, 1]$. Thus, if $H(x, w)$ is to be as close as possible to $G(x)$, then the network output should have a density that is close to uniform in $[0, 1]$. This is what our goal will be. We will attempt to train the network such that its output density is uniform, then the network mapping should represent the distribution function $G(x)$. The basic idea behind the proposed algorithm is to use the $N$ data points as inputs to the network. For every training cycle, we generate a different set of $N$ network targets randomly from a uniform distribution in $[0, 1]$, and adjust

the weights to map the data points (sorted in ascending order) to these generated targets (also sorted in ascending order). Thus we are training the network to map the data to a uniform distribution.

Before describing the steps of the algorithm, we note that the resulting network has to represent a monotonically nondecreasing mapping, otherwise it will not represent a legitimate distribution function. In our simulations, we used a hint penalty to enforce monotonicity [5]. The algorithm is as follows.

1. Let $x_1 \leq x_2 \leq \cdots \leq x_N$ be the data points. Set $t = 1$, where $t$ is the training cycle number. Initialize the weights (usually randomly) to $w(1)$.

2. Generate randomly from a uniform distribution in $[0, 1]$ $N$ points (and sort them): $u_1 \leq u_2 \leq ... \leq u_N$. The point $u_n$ is the target output for $x_n$.

3. Adjust the network weights according to the backpropagation scheme:

$$w(t + 1) = w(t) - \eta(t)\frac{\partial \mathcal{E}}{\partial w} \tag{1}$$

where $\mathcal{E}$ is the objective function that includes the error term and the monotonicity hint penalty term [5]:

$$\mathcal{E} = \sum_{n=1}^{N}\Big[H(x_n) - u_n\Big]^2 + \lambda \sum_{k=1}^{N_h}\Theta\Big(H(y_k) - H(y_k + \Delta)\Big)\Big[H(y_k) - H(y_k + \Delta)\Big]^2 \tag{2}$$

where we have suppressed the $w$ dependence. The second term is the monotonicity penalty term, $\lambda$ is a positive weighting constant, $\Delta$ is a small positive number, $\Theta(x)$ is the familiar unit step function, and the $y_k$'s are any set of points where we wish to enforce the monotonicity.

4. Set $t = t + 1$, and go to step 2. Repeat until the error is small enough. Upon convergence, the density estimate is the derivative of $H$.

Note that as presented, the randomly generated targets are different for every cycle, which will have a smoothing effect that will allow convergence to a truly uniform distribution. One other version, that we have implemented in our simulation studies, is to generate new targets after every fixed number $L$ of cycles, rather than every cycle. This will generally improve the speed of convergence as there is more "continuity" in the learning process. Also note that it is preferable to choose the activation function for the output node to be in the range of 0 to 1, to ensure that the estimate of the distribution function is in this range.

SLC is only applicable to estimating univariate densities, because, for the multivariate case, the nonlinear mapping $y = G(x)$ will not necessarily result in a uniformly distributed output $y$. Fortunately, many, if not the majority of problems encountered in practice are univariate. This is because multivariate problems, with even a modest number of dimensions, need a huge amount of data to obtain statistically accurate results. The next method, is applicable to the multivariate case as well.

## 2.2  SIC (Smooth Interpolation of the Cumulative)

Again, we have a multilayer network, to which we input the point $\mathbf{x}$, and the network outputs the estimate of the distribution function. Let $g(\mathbf{x})$ be the true density function, and let $G(\mathbf{x})$ be the corresponding distribution function. Let $\mathbf{x} = (x^1, ..., x^d)^T$. The distribution function is given by

$$G(\mathbf{x}) = \int_{-\infty}^{x^1} \cdots \int_{-\infty}^{x^d} g(\mathbf{x})dx^1 \cdots x^d, \tag{3}$$

a straightforward estimate of $G(\mathbf{x})$ could be the fraction of data points falling in the area of integration:

$$\hat{G}(\mathbf{x}) = \frac{1}{N}\sum_{n=1}^{N}\Theta(\mathbf{x} - \mathbf{x}_n), \qquad (4)$$

where $\Theta$ is defined as

$$\Theta(\mathbf{x}) = \begin{cases} 1 & \text{if } x^i \geq 0 \text{ for all } i = 1, \ldots, d, \\ 0 & \text{otherwise.} \end{cases}$$

The method we propose uses such an estimate for the target outputs of the neural network. The estimate given by (4) is discontinuous. The neural network method developed here provides a smooth, and hence more realistic estimate of the distribution function. The density can be obtained by differentiating the output of the network with respect to its inputs.

For the low-dimensional case, we can uniformly sample (4) using a grid, to obtain the examples for the network. Beyond two or three dimensions, this becomes computationally intensive. Alternatively, one could sample the input space randomly (using say a uniform distribution over the approximate range of $\mathbf{x}_n$'s), and for every point determine the network target according to (4). Another option is to use the data points themselves as examples. The target for a point $\mathbf{x}_m$ would then be

$$\hat{G}(x_m) = \frac{1}{N-1}\sum_{n=1,\ n\neq m}^{N}\Theta(x_m - x_n). \qquad (5)$$

We also use monotonicity as a hint to guide the training. Once training is performed, and $H(\mathbf{x}, w)$ approximates $G(\mathbf{x})$, the density estimate can be obtained as

$$\hat{g}(\mathbf{x}) = \frac{\partial^d H(\mathbf{x}, w)}{\partial x^1 \cdots \partial x^d}. \qquad (6)$$

## 3 Simulation Results

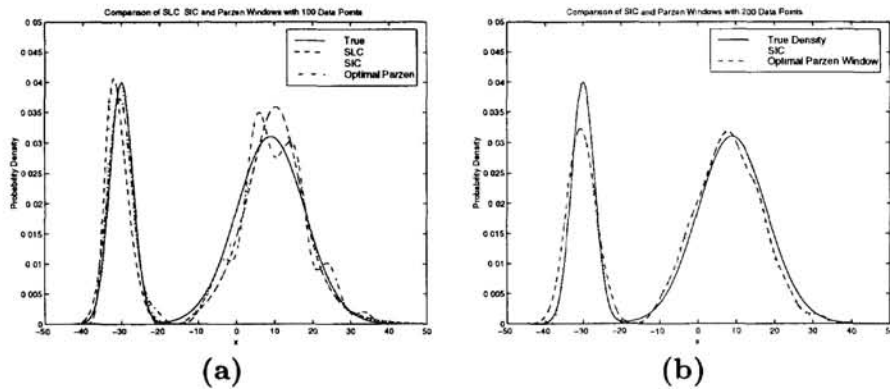

Figure 1: Comparison of optimal Parzen windows, with neural network estimators. Plotted are the true density and the estimates (SLC, SIC, Parzen window with optimal kernel width [6, pg 40]). Notice that even the optimal Parzen window is bumpy as compared to the neural network.

We tested our techniques for density estimation on data drawn from a mixture of two Gaussians:

$$g(x) = \frac{3}{10}\frac{1}{\sqrt{18\pi}}e^{-\frac{(x+30)^2}{18}} + \frac{7}{10}\frac{1}{\sqrt{162\pi}}e^{-\frac{(x-9)^2}{162}} \qquad (7)$$

Data points were randomly generated and the density estimates using SLC or SIC (for 100 and 200 data points) were compared to the Parzen technique. Learning was performed with a standard 1 hidden layer neural network with 3 hidden units. The hidden unit activation function used was *tanh* and the output unit was an *erf* function[1]. A set of typical density estimates are shown in figure 1.

# 4   Convergence of the Density Estimation Techniques

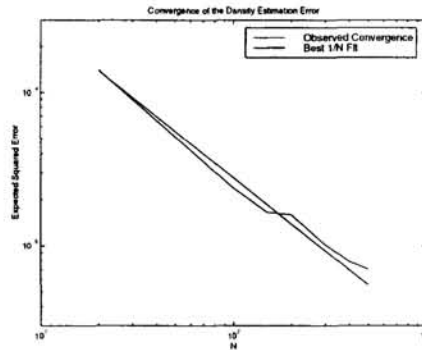

Figure 2: Convergence of the density estimation error for SIC. A five hidden unit two layer neural network was used to perform the mapping $x_i \to i/(N+1)$, trained according to SIC. For various $N$, the resulting density estimation error was computed for over 100 runs. Plotted are the results on a Log-Log scale. For comparison, also shown is the best $1/N$ fit.

Using techniques from stochastic approximation theory, it can be shown that SLC converges to a similar solution to SIC [3], so, we focus our attention on the convergence of SIC. Figure 2 shows an empirical study of the convergence behavior. The optimal linear fit between $\log(E)$ and $\log(N)$ has a slope of $-0.97$. This indicates that the convergence rate is about $1/N$. The theoretically derived convergence rate is $\log\log(N)/N$ as we will shortly discuss.

To analyze SIC, we introduce so called approximate generalized distribution functions. We will assume that the true distribution function has bounded derivatives. Therefore the cumulative will be "approximately" implementable by generalized distributions with bounded derivatives (in the asymptotic limit, with probability 1). We will then obtain the convergence to the true density.

Let $\mathcal{G}$ be the space of distribution functions on the real line that possess continuous densities, i.e., $X \in \mathcal{G}$ if $X : \mathbf{R} \to [0,1]$; $X'(t)$ exists everywhere, is continuous and $X'(t) \geq 0$; $X(-\infty) = 0$ and $X(\infty) = 1$. This is the class of functions that we will be interested in. We define a metric with respect to $\mathcal{G}$ as follows

$$\| f \|_X^2 = \int_{-\infty}^{\infty} f(t)^2 X'(t) dt \qquad (8)$$

$\| f \|_X^2$ is the expectation of the squared value of $f$ with respect to the distribution $X \in \mathcal{G}$. Let us name this the $L_2$ $X$-norm of $f$. Let the data set $(D)$ be $\{x_1 \leq x_2 \leq \ldots \leq x_N\}$, and corresponding to each $x_i$, let the target be $y_i = i/N + 1$. We will assume that the true distribution function has bounded derivatives up to order $K$. We define the set of approximate sample distribution functions $\mathcal{H}_D^\nu$ as follows

**Definition 4.1** *Fix $\nu > 0$. A $\nu$-approximate sample distribution function, $H$, satisfies the following two conditions*

- $H \in \mathcal{G}$

- $|H(x_i) - y_i| \leq \nu \sqrt{\frac{\log\log(N)}{2N}}, \ \forall i$

*We will denote the set of all $\nu$-approximate sample distribution functions for a data set, $D$, and a given $\nu$ by $\mathcal{H}_D^\nu$.*

Let $A_i = \sup_x |G^{(i)}|$, $i = 1 \ldots K$ where we use the notation $f^{(i)}$ to denote the $i^{th}$ derivative. Define $B_i^\nu(D)$ by

$$B_i^\nu(D) = \inf_{Q \in \mathcal{H}_D^\nu} \sup_x |Q^{(i)}| \tag{9}$$

for fixed $\nu > 0$. Note that by definition, for all $\epsilon > 0$, $\exists \ H \in \mathcal{H}_D^\nu$ such that $\sup_x |H^{(i)}(x)| \leq B_i^\nu + \epsilon$. $B_i^\nu(D)$ is the lowest possible bound on the $i^{th}$ derivative for the $\nu$-approximate sample distribution functions given a particular data set. In a sense, the "smoothest" approximating sample distribution function with respect to the $i^{th}$ derivative has an $i^{th}$ derivative bounded by $B_i^\nu(D)$. One expects that $B_i \leq A_i$, at least in the limit $N \to \infty$.

In the next theorem, we present the main theoretical result of the paper, namely a bound on the estimation error for the density estimator obtained by using the approximate sample distribution functions. It is embedded in a large amount of technical machinery, but its essential content is that if the true distribution function has bounded derivatives to order $K$, then, picking the approximate distribution function obeying certain bounds, we obtain a convergence rate for the estimation error of $O((\log\log(N)/N)^{1-1/K})$.

**Theorem 4.2 ($L_2$ convergence to the true density)** *Let $N$ data points, $x_i$ be drawn i.i.d. from the distribution $G \in \mathcal{G}$. Let $\sup_x |G^{(i)}| = A_i$ for $i = 0 \ldots K$, where $K \geq 2$. Fix $\nu > 2$ and $\epsilon > 0$. Let $B_K^\nu(D) = \inf_{Q \in \mathcal{H}_D^\nu} \sup_x |Q^{(K)}|$. Let $H \in \mathcal{H}_D^\nu$ be a $\nu$-approximate distribution function with $B_K = \sup_x |H^K| \leq B_K^\nu + \epsilon$ (by the definition of $B_K^\nu$, such a $\nu$-approximate sample distribution function must exist). Then, for any $F \in \mathcal{G}$, as $N \to \infty$, the inequality*

$$\| H' - G' \|_F^2 \leq 2^{2(K-1)} (2A_K + \epsilon)^{\frac{2}{K}} \mathcal{F}(N) \tag{10}$$

*where*

$$\mathcal{F}(N) = \left[ (1 + \nu) \left( \frac{2\log\log(N)}{N} \right)^{\frac{1}{2}} + \frac{2}{N+1} \right]^{2 - \frac{2}{K}} \tag{11}$$

*holds with probability 1, as $N \to \infty$.*

We present the proof elsewhere [3]. ∎

**Note 1:** The theorem applies uniformly to *any* interpolator $H \in \mathcal{H}_D^\nu$. In particular, a large enough neural network will be one such monotonic interpolator, provided that the network can be trained to small enough error. This is possible by the universal approximation results for multilayer networks [2].

**Note 2:** This theorem holds for any $\epsilon > 0$ and $\nu > 1$. For smooth density functions, with bounded higher derivatives, the convergence rate approaches $O(\log\log(N)/N)$ which is faster convergence than the kernel density estimator (for which the optimal rate is $O(N^{-4/5})$).

**Note 3:** No smoothing parameter needs to be determined.

**Note 4:** One should try to find an approximate distribution function with the smallest possible derivatives. Specifically, of all the sample distribution functions, pick the one that "minimizes" $B_K$, the bound on the $K^{th}$ derivative. This could be done by introducing penalty terms, penalizing the magnitudes of the derivatives (for example Tikhonov type regularizers [7]).

## 5  Comments

We developed two techniques for density estimation based on the idea of learning the cumulative by mapping the data points to a uniform density. Two techniques were presented, a stochastic technique (SLC), which is expected to inherit the characteristics of most stochastic iterative algorithms, and a deterministic technique (SIC). SLC tends to be slow in practice, however, because each set of targets is drawn from the uniform distribution, this is anticipated to have a smoothing/regularizing effect – this can be seen by comparing SLC and SIC in figure 1 (a). We presented experimental comparison of our techniques with the Parzen technique.

We presented a theoretical result that demonstrated the consistency of our techniques as well as giving a convergence rate of $O(\log\log(N)/N)$, which is better than the optimal Parzen technique. No smoothing parameter needs to be chosen – smoothing occurs naturally by picking the interpolator with the lowest bound for a certain derivative. For our methods, the majority of time is spent in the learning phase, but once learning is done, evaluating the density is fast.

## 6  Acknowledgments

We would like to acknowledge Yaser Abu-Mostafa and the Caltech Learning Systems Group for their useful input.

## Footnotes

*To whom correspondence should be addressed.

[1] $erf(x) = \frac{1}{\sqrt{2\pi}} \int_{-\infty}^{x} e^{-\frac{x^2}{2}}.$

## References

[1] K. Fukunaga and L. D. Hostetler. Optimization of $k$-nearest neighbor density estimates. *IEEE Transactions on Information Theory*, 19(3):320–326, 1973.

[2] K. Hornik, M. Stinchcombe, and H. White. Universal approximation of an unknown mapping and its derivatives using multilayer feedforward networks. *Neural Networks*, 3:551–560, 1990.

[3] M. Magdon-Ismail and A. Atiya. Consistent density estimation from the sample distribution function. *manuscript in preparation for submission*, 1998.

[4] E. Parzen. On the estimation of a probability density function and mode. *Annals of Mathematical Statistics*, 33:1065–1076, 1962.

[5] J. Sill and Y. S. Abu-Mostafa. Monotonicity hints. In M. C. Mozer, M. I. Jordan, and T. Petsche, editors, *Advances in Neural Information Processing Systems (NIPS)*, volume 9, pages 634–640. Morgan Kaufmann, 1997.

[6] B. Silverman. *Density Estimation for Statistics and Data Analysis*. Chapman and Hall, London, UK, 1993.

[7] A. N. Tikhonov and V. I. Arsenin. *Solutions of Ill-Posed Problems*. Scripta Series in Mathematics. Distributed solely by Halsted Press, Winston; New York, 1977. Translation Editor: Fritz, John.
